# Dynamical Constraints on Computing with Spike Timing in the Cortex

**Arunava Banerjee** and **Alexandre Pouget**
Department of Brain and Cognitive Sciences
University of Rochester, Rochester, New York 14627
*{arunavab,alex}@bcs.rochester.edu*

## Abstract

If the cortex uses spike timing to compute, the timing of the spikes must be robust to perturbations. Based on a recent framework that provides a simple criterion to determine whether a spike sequence produced by a generic network is sensitive to initial conditions, and numerical simulations of a variety of network architectures, we argue within the limits set by our model of the neuron, that it is unlikely that precise sequences of spike timings are used for computation under conditions typically found in the cortex.

## 1 Introduction

Several models of neural computation use the precise timing of spikes to encode information. For example, Abeles et al. have proposed synchronous volleys of spikes (synfire chains) as a candidate for representing information in the cortex [1]. More recently, Maass has demonstrated how spike timing in general, not merely synfire chains, can be utilized to perform nonlinear computations [6].

For any of these schemes to function, the timing of the spikes must be robust to small perturbations; i.e., small perturbations of spike timing should not result in successively larger fluctuations in the timing of subsequent spikes. To use the terminology of dynamical systems theory, the network must not exhibit sensitivity to initial conditions. Indeed, reliable computation would simply be impossible if the timing of spikes is sensitive to the slightest source of noise, such as synaptic release variability, or thermal fluctuations in the opening and closing of ionic channels.

Diesmann et al. have recently examined this issue for the particular case of synfire chains in feed-forward networks [4]. They have demonstrated that the propagation of a synfire chain over several layers of integrate-and-fire neurons can be robust to 2 Hz of random background activity and to a small amount of noise in the spike timings. The question we investigate here is whether this result generalizes to the propagation of any arbitrary spatiotemporal configuration of spikes through a recurrent network of neurons. This question is central to any theory of computation in cortical networks using spike timing since it is well known that the connectivity between neurons in the cortex is highly recurrent. Although there have been earlier attempts at resolving like issues, the applicability of the results are limited by the model of the neuron [8] or the pattern of propagated spikes [5] considered.

Before we can address this question in a principled manner, however, we must confront a couple of confounding issues. First stands the problem of stationarity. As is well known, Lyapunov characteristic exponents of trajectories are limit quantities that are guaranteed to exist (almost surely) in classical dynamical systems that are stationary. In systems such as the cortex that receive a constant barrage of transient inputs, it is questionable whether such a concept bears much relevance. Fortunately, our simulations indicate that convergence or divergence of trajectories in cortical networks can occur very rapidly (within 200-300 msec). Assuming that external inputs do not change drastically over such short time scales, one can reasonably apply the results from analysis under stationary conditions to such systems.

Second, the issues of how a network should be constructed so as to generate a particular spatiotemporal pattern of spikes as well as whether a given spatiotemporal pattern of spikes can be generated in principle, remain unresolved in the general setting. It might be argued that without such knowledge, any classification of spike patterns into sensitive and insensitive classes is inherently incomplete. However, as shall be demonstrated later, sensitivity to initial conditions can be inferred under relatively weak conditions. In addition, we shall present simulation results from a variety of network architectures to support our general conclusions.

The remainder of the paper is organized as follows. In section 2, we briefly review relevant aspects of the dynamical system corresponding to a recurrent neuronal network as formulated in [2] and formally define "sensitivity to initial conditions". In Section 3, we present simulation results from a variety of network architectures. In Section 4, we interpret these results formally which in turn lead us to an additional set of experiments. In Section 5, we draw conclusions regarding the issue of computation using spike timing in cortical networks based on these results.

## 2 Spike dynamics

A detailed exposition of an abstract dynamical system that models recurrent systems of biological neurons was presented in [2]. Here, we recount those aspects of the system that are relevant to the present discussion. Based on the intrinsic nature of the processes involved in the generation of postsynaptic potentials (PSP's) and of those involved in the generation of action potentials (spikes), it was shown that the state of a system of neurons can be specified by enumerating the temporal positions of all spikes generated in the system over a bounded past. For example, in Figure 1, the present state of the system is described by the positions of the spikes (solid lines) in the shaded region at $t=0$ and the state of the system at a future time $T$ is specified by the positions of the spikes (solid lines) in the shaded region at $t=T$. Each internal neuron $i$ in the system is assigned a membrane potential function $P_i(\cdot)$ that takes as its input the present state and generates the instantaneous potential at the soma of neuron $i$. It is the particular instantiation of the set of functions $P_i(\cdot)$ that determines the nature of the neurons as well as their connectivity in the network.

Consider now the network in Figure 1 initialized at the particular state described by the shaded region at $t=0$. Whenever the integration of the PSP's from all presynaptic spikes to a neuron combined with the hyperpolarizing effects of its own spikes (the precise nature of the union specified by $P_i(\cdot)$) brings its membrane potential above threshold, the neuron emits a new spike. If the spikes in the shaded region at $t=0$ were perturbed in time (dotted lines), this would result in a perturbation on the new spike. The size of the new perturbation would depend upon the positions of the spikes in the shaded region, the nature of $P_i(\cdot)$, and the sizes of the old perturbations. This scenario would in turn repeat to produce further perturbations on future spikes. In essence, any initial set of perturbations would propagate from spike to spike to produce a set of perturbations at any arbitrary future time $t=T$.

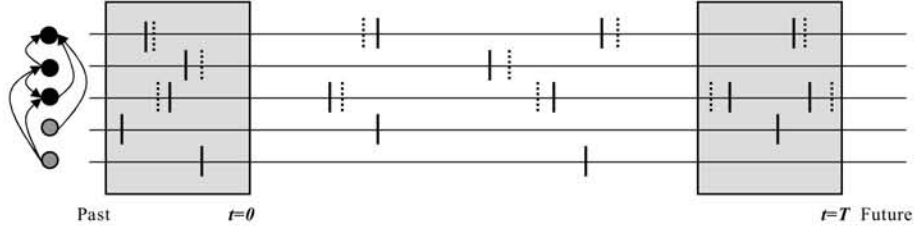

Past           *t=0*                                         *t=T*   Future

Figure 1: Schematic diagram of the spike dynamics of a system of neurons. Input neurons are colored gray and internal neurons black. Spikes are shown in solid lines and their corresponding perturbations in dotted lines. Note that spikes generated by the input neurons are not perturbed. Gray boxes demarcate a bounded past history starting at time *t*. The temporal position of all spikes in the boxes specify the state of the system at times *t=0* and *t=T*.

It is of considerable importance to note at this juncture that while the specification of the network architecture and the synaptic weights determine the precise temporal sequence of spikes generated by the network, the relative size of successive perturbations are determined by the temporal positions of the spikes in successive state descriptions at the instant of the generation of each new spike. If it can be demonstrated that there are particular classes of state descriptions that lead to large relative perturbations, one can deduce the qualitative aspects of the dynamics of a network armed with only a general description of its architecture. A formal analysis in Section 4 will bring to light such a classification.

Let column vectors $\vec{x}$ and $\vec{y}$ denote, respectively, perturbations on the spikes of internal neurons at times *t=0* and *t=T*. We pad each vector with as many zeroes as there are input spikes in the respective state descriptions. Let $A_T$ denote the matrix such that $\vec{y} = A_T \vec{x}$. Let $B$ and $C$ be the matrices as described in [3] that discard the rigid translational components from the final and initial perturbations. Then, the dynamics of the system is sensitive to initial conditions if $\lim_{T \to \infty} \|B * A_T * C\| = \infty$.

If instead, $\lim_{T \to \infty} \|B * A_T * C\| = 0$, the dynamics is insensitive to initial conditions.

A few comments are in order here. First, our interest lies not in the precise values of the Lyapunov characteristic exponents of trajectories (where they exist), but in whether the largest exponent is greater than or less than zero. Furthermore, the class of trajectories that satisfy either of the above criteria is larger (although not necessarily in measure) than the class of trajectories that have definite exponents. Second, input spikes are free parameters that have to be constrained in some manner if the above criteria are to be well-defined. By the same token, we do not consider the effects that perturbations of input spikes have on the dynamics of the system.

## 3   Simulations and results

A typical column in the cortex contains on the order of $10^5$ neurons, approximately 80% of which are excitatory and the rest inhibitory. Each neuron receives around $10^4$ synapses, approximately half of which are from neurons in the same column and the rest from excitatory neurons in other columns and the thalamus. These estimates indicate that even at background rates as low as 0.1 Hz, a column generates on average 10 spikes every millisecond. Since perturbations are propagated from spikes

to generated spikes, divergence and/or convergence of spike trajectories could occur extremely rapidly. We test this hypothesis in this section through model simulations.

All experiments reported here were conducted on a system containing 1000 internal neurons (set to model a cortical column) and 800 excitatory input neurons (set to model the input into the column). Of the 1000 internal neurons, 80% were chosen to be excitatory and the rest inhibitory. Each internal neuron received 100 synapses from other (internal as well as input) neurons in the system. The input neurons were set to generate random uncorrelated Poisson spike trains at a fixed rate of 5 Hz.

The membrane potential function $P_i(\cdot)$ for each internal neuron was modeled as the sum of excitatory and inhibitory PSP's triggered by the arrival of spikes at synapses, and afterhyperpolarization potentials triggered by the spikes generated by the neuron. PSP's were modeled using the function $\frac{\omega}{\nu\sqrt{t}}e^{-\varepsilon\nu^2/t}e^{-t/\tau}$ where $\nu$, $\varepsilon$ and $\tau$ were set to mimic four kinds of synapses, NMDA, AMPA, GABA$_A$, and GABA$_B$. $\omega$ was set for excitatory and inhibitory synapses so as to generate a mean spike rate of 5 Hz by excitatory and 15 Hz by inhibitory internal neurons. The parameters were then held constant over the entire system leaving the network connectivity and axonal delays as the only free parameters. After the generation of a spike, an absolute refractory period of 1 msec was introduced during which the neuron was prohibited from generating a spike. There was no voltage reset. However, each spike triggered an afterhyperpolarization potential with a decay constant of 30 msec that led to a relative refractory period. Simulations were performed in 0.1 msec time steps and the time bound on the state description, as related in Section 2, was set at 200 msec.

The issue of correlated inputs was addressed by simulating networks of disparate architectures. On the one extreme was an ordered two layer ring network with input neurons forming the lower layer and internal neurons (with the inhibitory neurons placed evenly among the excitatory neurons) forming the upper layer. Each internal neuron received inputs from a sector of internal and input neurons that was centered on that neuron. As a result, any two neighboring internal neurons shared 96 of their 100 inputs (albeit with different axonal delays of 0.5-1.1 msec). This had the effect of output spike trains from neighboring internal neurons being highly correlated, with sectors of internal neurons producing synchronized bursts of spikes. On the other extreme was a network where each internal neuron received inputs from 100 randomly chosen neurons from the entire population of internal and input neurons. Several other networks where neighboring internal neurons shared an intermediate percentage of their inputs were also simulated. Here, we present results from the two extreme architectures. The results from all the other networks were similar.

Figure 2(a) displays sample output spike trains from 100 neighboring internal neurons over a period of 450 msec for both architectures. In the first set of experiments, pairs of identical systems driven by identical inputs and initialized at identical states except for one randomly chosen spike that was perturbed by 1 msec, were simulated. In all cases, the spike trajectories diverged very rapidly. Figure 2(b) presents spike trains generated by the same 100 neighboring internal neurons from the two simulations from 200 to 400 msec after initialization, for both architectures.

To further explore the sensitivity of the spike trajectories, we partitioned each trajectory into segments of 500 spike generations each. For each such segment, we then extracted the spectral norm $\lVert B * A_T * C \rVert$ after every 100 spike generations.

Figure 2(c) presents the outcome of this analysis for both architectures. Although successive segments of 500 spike generations were found to be quite variable in their absolute sensitivity, each such segment was nevertheless found to be sensitive. We also simulated several other architectures (results not shown), such as systems with fixed axonal delays and ones with bursty behavior, with similar outcomes.

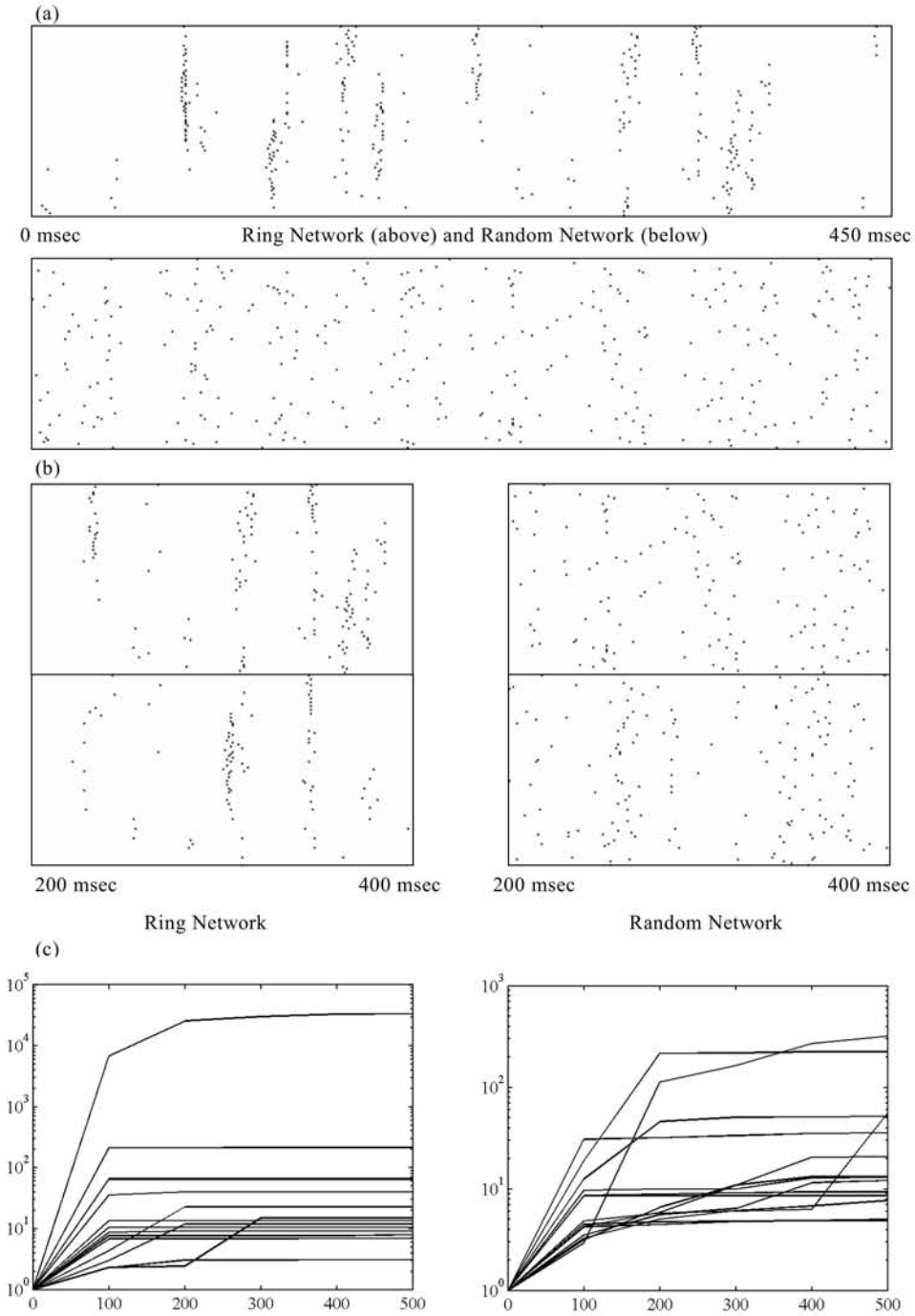

Figure 2: (a) Spike trains of 100 neighboring neurons for 450 msec from the ring and the random networks respectively. (b) Spike trains from the same 100 neighboring neurons (above and below) 200 msec after initialization. Note that the trains have already diverged at 200 msec. (c) Spectral norm of sensitivity matrices of 14 successive segments of 500 spike generations each, computed in steps of 100 spike generations for both architectures.

# 4  Analysis and further simulations

The reasons behind the divergence of the spike trajectories presented in Section 3 can be found by considering how perturbations are propagated from the set of spikes in the current state description to a newly generated spike. As shown in [3], the perturbation in the new spike can be represented as a weighted sum of the perturbations of those spikes in the state description that contribute to the generation of the new spike. The weight assigned to a spike $x_i$ is proportional to the slope of the PSP or that of the hyperpolarization triggered by that spike ($\partial P(\cdot)/\partial x_i$ in the general case), at the instant of the generation of the new spike. Intuitively, the larger the slope is, the greater is the effect that a perturbation of that spike can have on the total potential at the soma, and hence, the larger is the perturbation on the new spike. The proportionality constant is set so that the weights sum to 1. This constraint is reflected in the fact that if all spikes were to be perturbed by a fixed quantity, this would amount to a rigid displacement in time causing the new spike to be perturbed by the same quantity. We denote the slopes by $\rho_i$, and the weights by $\alpha_i$. Then, $\alpha_i = \rho_i \big/ \sum_{j=1}^{n} \rho_j$, where $j$ ranges over all contributing spikes.

We now assume that at the generation of each new spike, the $\rho_t$'s are drawn independently from a stationary distribution (for both internal and input contributing spikes), and that the ratio of the number of internal to the total (internal plus input) spikes in any state description remains close to a fixed quantity $\mu$ at all times. Note that this amounts to an assumed probability distribution on the likelihood of particular spike trajectories rather than one on possible network architectures and synaptic weights. The iterative construction of the matrix $A_T$, based on these conditions, was described in detail in [3]. It was also shown that the statistic $\left\langle \sum_{i=1}^{n} \alpha_i^2 \right\rangle$ plays a central role in the determination of the sensitivity of the resultant spike trajectories. In a minor modification to the analysis in [3], we assume that $A_T$ represents the full perturbation (internal plus input) at each step of the process. While this merely entails the introduction of additional rows with zero entries to account for input spikes in each state, this alters the effect that $B$ has on $\lVert B * A_T * C \rVert$ in a way that allows for a simpler as well as bidirectional bound on the norm. Since the analysis is identical to that in [3] and does not introduce any new techniques, we only report the result. If $\left\langle \sum_{i=1}^{n} \alpha_i^2 \right\rangle > \dfrac{(2 + O(1/m))}{\mu} - 1$ $\left( \text{resp. } \left\langle \sum_{i=1}^{n} \alpha_i^2 \right\rangle < \dfrac{1}{\mu} - 1 \right)$, then the spike trajectories are almost surely sensitive (resp. insensitive) to initial conditions. $m$ denotes the number of internal spikes in the state description.

If we make the liberal assumption that input spikes account for as much as half the total number of spikes in state descriptions, noting that $m$ is a very large quantity (greater than $10^3$ in all our simulations), the above constraint requires $\left\langle \sum \alpha_i^2 \right\rangle > 3$ for spike trajectories to be almost surely sensitive to initial conditions. From our earlier simulations, we extracted the value of $\sum \alpha_i^2$ whenever a spike was generated, and computed the sample mean $\left\langle \sum \alpha_i^2 \right\rangle$ over all spike generations. The mean was larger than 3 in all cases (it was 69.6 for the ring and 11.3 for the random network).

The above criterion enables us to peer into the nature of the spike dynamics of real cortical columns, for although simulating an entire column remains intractable, a single neuron can be simulated under various input scenarios, and the resultant statistic applied to infer the nature of the spike dynamics of a cortical column most of whose neurons operate under those conditions.

An examination of the mathematical nature of $\sum \alpha_i^2$ reveals that its value rises as the size of the subset of $\rho_i$'s that are negative grows larger. The criterion for sensitivity is therefore more likely to be met when a substantial portion of the excitatory PSP's are on their falling phase (and inhibitory PSP's on their rising phase) at the instant of the generation of each new spike. This corresponds to a case where the inputs into the neurons of a system are not strongly synchronized. Conversely, if spikes are generated soon after the arrival of a synchronized burst of spikes (all of whose excitatory PSP's are presumably on their rising phase), the criterion for sensitivity is less likely to be met. We simulated several combinations of the two input scenarios to identify cases where the corresponding spike trajectories in the system were not likely to be sensitive to initial conditions.

We constructed a model pyramidal neuron with 10,000 synapses, 85% of which were chosen to be excitatory and the rest inhibitory. The threshold of the neuron was set at 15 mV above resting potential. PSP's were modeled using the function described earlier with values for the parameters set to fit the data reported in [7]. For excitatory PSP's the peak amplitudes ranged between 0.045 and 1.2 mV with the median around 0.15 mV, 10-90 rise times ranged from 0.75 to 3.35 msec and widths at half amplitude ranged from 8.15 to 18.5 msec. For inhibitory PSP's, the peak amplitudes were on average twice as large and the 10-90 rise times and widths at half amplitude were slightly larger. Whenever the neuron generated a new spike, the values of the $\rho_i$'s were recorded and $\sum \alpha_i^2$ was computed. The mean $\left\langle \sum \alpha_i^2 \right\rangle$ was then computed over the set of all spike generations. In order to generate conservative estimates, samples with value above $10^4$ were discarded (they comprised about 0.1% of the data). The datasets ranged in size from 3000 to 15,000.

Three experiments simulating various levels of uncorrelated input/output activity were conducted. In particular, excitatory Poisson inputs at 2, 20 and 40 Hz were balanced by inhibitory Poisson inputs at 6.3, 63 and 124 Hz to generate output rates of approximately 2, 20 and 40 Hz, respectively. We confirmed that the output in all three cases was Poisson-like (CV=0.77, 0.74, and 0.89, respectively). The mean $\left\langle \sum \alpha_i^2 \right\rangle$ for the three experiments was 4.37, 5.66, and 9.52, respectively.

Next, two sets of experiments simulating the arrival of regularly spaced synfire chains were conducted. In the first set the random background activity was set at 2 Hz and in the second, at 20 Hz. The synfire chains comprised of spike volleys that arrived every 50 msec. Four experiments were conducted within each set: volleys were composed of either 100 or 200 spikes (producing jolts of around 10 and 20 mV respectively) that were either fully synchronized or were dispersed over a Gaussian distribution with σ=1 msec. The mean $\left\langle \sum \alpha_i^2 \right\rangle$ for the experiments was as follows.

At 2 Hz background activity, it was 0.49 (200 spikes/volley, synchronized), 0.60 (200 spikes/volley, dispersed), 2.46 (100 spikes/volley, synchronized), and 2.16 (100 spikes/volley, dispersed). At 20 Hz background activity, it was 4.39 (200 spikes/volley, synchronized), 8.32 (200 spikes/volley, dispersed), 6.77 (100 spikes/volley, synchronized), and 6.78 (100 spikes/volley, dispersed).

Finally, two sets of experiments simulating the arrival of randomly spaced synfire chains were conducted. In the first set the random background activity was set at 2 Hz and in the second, at 20 Hz. The synfire chains comprised of a sequence of spike volleys that arrived randomly at a rate of 20 Hz. Two experiments were conducted within each set: volleys were composed of either 100 or 200 synchronized spikes. The mean $\left\langle \sum \alpha_i^2 \right\rangle$ for the experiments was as follows. At 2 Hz background activity, it was 4.30 (200 spikes/volley) and 4.64 (100 spikes/volley). At 20 Hz background activity, it was 5.24 (200 spikes/volley) and 6.28 (100 spikes/volley).

# 5 Conclusion

As was demonstrated in Section 3, sensitivity to initial conditions transcends unstructured connectivity in systems of spiking neurons. Indeed, our simulations indicate that sensitivity is more the rule than the exception in systems modeling cortical networks operating at low to moderate levels of activity. Since perturbations are propagated from spike to spike, trajectories that are sensitive can diverge very rapidly in systems that generate a large number of spikes within a short period of time. Sensitivity therefore is an issue, even for schemes based on precise sequences of spike timing with computation occurring over short (hundreds of msec) intervals.

Within the limits set by our model of the neuron, we have found that spike trajectories are likely to be sensitive to initial conditions in all scenarios except where large (100-200) synchronized bursts of spikes occur in the presence of sparse background activity (2 Hz) with sufficient but not too large an interval between successive bursts (50 msec). This severely restricts the possible use of precise spike sequences for reliable computation in cortical networks for at least two reasons. First, unsynchronized activity can rise well above 2 Hz in the cortex, and second, the highly constrained nature of this dynamics would show in *in vivo* recordings.

Although cortical neurons can have vastly more complex responses than that modeled in this paper, our conclusions are based largely on the simplicity and the generality of the constraints identified (the analysis assumes a general membrane potential function $P(\cdot)$). Although a more refined model of the cortical neuron could lead to different values of the statistic computed, we believe that the results are unlikely to cross the noted bounds and therefore change our overall conclusions.

We are however not arguing that computation with spike timing is impossible in general. There are neural structures, such as the nucleus laminaris in the barn owl and the electrosensory array in the electric fish, which have been shown to perform exquisitely precise computations using spike timing. Interestingly, these structures have very specialized neurons and network architectures.

To conclude, computation using precise spike sequences does not appear to be likely in the cortex in the presence of Poisson-like activity at levels typically found there.

## References

[1] Abeles, M., Bergman, H., Margalit, E. & Vaadia, E. (1993) Spatiotemporal firing patterns in the frontal cortex of behaving monkeys. *Journal of Neurophysiology* **70**, pp. 1629-1638.

[2] Banerjee, A. (2001) On the phase-space dynamics of systems of spiking neurons: I. model and experiments. *Neural Computation* **13,** pp. 161-193.

[3] Banerjee, A. (2001) On the phase-space dynamics of systems of spiking neurons: II. formal analysis. *Neural Computation* **13**, pp. 195-225.

[4] Diesmann, M., Gewaltig, M. O. & Aertsen, A. (1999) Stable propagation of synchronous spiking in cortical neural networks. *Nature* **402**, pp. 529-533.

[5] Gerstner, W., van Hemmen, J. L. & Cowan, J. D. (1996) What matters in neuronal locking. *Neural Computation* **8**, pp. 1689-1712.

[6] Maass, W. (1995) On the computational complexity of networks of spiking neurons. *Advances in Neural Information Processing Systems 7*, pp. 183-190.

[7] Mason, A., Nicoll, A. & Stratford, K. (1991) Synaptic transmission between individual pyramidal neurons of the rat visual cortex in vitro. *Journal of Neuroscience* **11**(1), pp. 72-84.

[8] van Vreeswijk, C., & Sompolinsky, H. (1998) Chaotic balanced state in a model of cortical circuits. *Neural Computation* **10**, pp. 1321-1372.
